# The Clusteron: Toward a Simple Abstraction for a Complex Neuron

**Bartlett W. Mel**
Computation and Neural Systems
Division of Biology
Caltech, 216-76
Pasadena, CA 91125
mel@cns.caltech.edu

## Abstract

Are single neocortical neurons as powerful as multi-layered networks? A recent compartmental modeling study has shown that voltage-dependent membrane nonlinearities present in a complex dendritic tree can provide a virtual layer of local nonlinear processing elements between synaptic inputs and the final output at the cell body, analogous to a hidden layer in a multi-layer network. In this paper, an abstract model neuron is introduced, called a *clusteron*, which incorporates aspects of the dendritic "cluster-sensitivity" phenomenon seen in these detailed biophysical modeling studies. It is shown, using a *clusteron*, that a Hebb-type learning rule can be used to extract higher-order statistics from a set of training patterns, by manipulating the spatial ordering of synaptic connections onto the dendritic tree. The potential neurobiological relevance of these higher-order statistics for nonlinear pattern discrimination is then studied within a full compartmental model of a neocortical pyramidal cell, using a training set of 1000 high-dimensional sparse random patterns.

## 1   INTRODUCTION

The nature of information processing in complex dendritic trees has remained an open question since the origin of the neuron doctrine 100 years ago. With respect to learning, for example, it is not known whether a neuron is best modeled as a pseudo-linear unit, equivalent in power to a simple Perceptron, or as a general nonlinear learning device, equivalent in power to a multi-layered network. In an attempt to characterize the input-output behavior of a whole dendritic tree containing voltage-dependent membrane mechanisms, a recent compartmental modeling study in an anatomically reconstructed neocortical pyramidal cell (anatomical data from Douglas et al., 1991; "NEURON" simulation package provided by Michael Hines and John Moore) showed that a dendritic tree rich in NMDA-type synaptic channels is selectively responsive to spatially clustered, as opposed to diffuse, pattens of synaptic activation (Mel, 1992). For example, 100 synapses which were simultaneously activated at 100 randomly chosen locations about the dendritic arbor were less effective at firing the cell than 100 synapses activated in groups of 5, at each of 20 randomly chosen dendritic locations. The cooperativity among the synapses in each group is due to the voltage dependence of the NMDA channel: Each activated NMDA synapse becomes up to three times more effective at injecting synaptic current when the post-synaptic membrane is locally depolarized by 30-40 mV from the resting potential. When synapses are activated in a group, the depolarizing effects of each helps the others (and itself) to move into this more efficient voltage range.

This work suggested that the spatial *ordering* of afferent synaptic connections onto the dendritic tree may be a crucial determinant of cell responses to specific input patterns. The nonlinear interactions among neighboring synaptic inputs further lent support to the idea that two or more afferents that form closely grouped synaptic connections on a dendritic tree may be viewed as encoding higher-order input-space "features" to which the dendrite is sensitive (Feldman & Ballard, 1982; Mel, 1990; Durbin & Rumelhart, 1990). The more such higher-order features are present in a given input pattern, the more the spatial distribution of active synapses will be clustered, and hence the more the post-synaptic cell will be inclined to fire in response. In a demonstration of this idea through direct manipulation of synaptic ordering, dendritic cluster-sensitivity was shown to allow the model neocortical pyramidal cell to reliably discriminate 50 training images of natural scenes from untrained control images (see Mel, 1992). Since all presented patterns activated the same number of synapses of the same strength, and with no systematic variation in their dendritic locations, the underlying dendritic "discriminant function" was necessarily nonlinear.

A crucial question remains as to whether other, e.g. non-synaptic, membrane nonlinearities, such as voltage-dependent calcium channels in the dendritic shaft membrane, could enhance, abolish, or otherwise alter the dendritic cluster-sensitivity phenomenon seen in the NMDA-only case. Some of the simulations presented in the remainder of this paper include voltage-dependent calcium channels and/or an anomalous rectification in the dendritic membrane. However, detailed discussions of these channels and their effects will be presented elsewhere.

## 2   THE CLUSTERON

### 2.1   MOTIVATION

This paper deals primarily with an important extension to the compartmental modeling experiments and the hand-tuned demonstrations of nonlinear pattern discrimi-

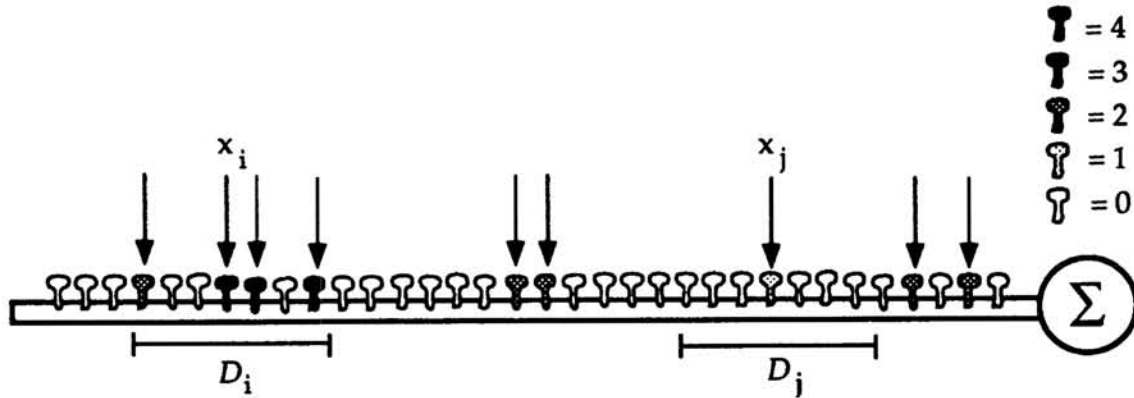

Figure 1: The *Clusteron*. Active inputs lines are designated by arrows; shading of synapses reflects synaptic activation $a_i$ when $x_i \in \{0, 1\}$ and weights are set to 1.

nation capacity presented in (Mel, 1992). If the manipulation of synaptic ordering is necessary for neurons to make effective use of their cluster-sensitive dendrites, then a learning mechanism capable of appropriately manipulating synaptic ordering must also be present in these neurons. An abstract model neuron called a *clusteron* is presented here, whose input-output relation was inspired by the idea of dendritic cluster-sensitivity, and whose learning rule is a variant of simple Hebbian learning. The *clusteron* is a far simpler and more convenient model for the study of cluster-sensitive learning than the full-scale compartmental model described in (Mel, 1992), whose solutions under varying stimulus conditions are computed through numerical integration of a system of several hundred coupled nonlinear differential equations (Hines, 1989). However, once the basic mathematical and algorithmic issues have been better understood, more biophysically detailed models of this type of learning in dendritic trees, as has been reported in (Brown et al., 1990), will be needed.

## 2.2 INPUT-OUTPUT BEHAVIOR

The *clusteron* is a particular second-order generalization of the thresholded linear unit (TLU), exemplified by the common Perceptron. It consists of a "cell body" where the globally thresholded output of the unit is computed, and a dendritic tree, which for present purposes will be visualized as a single long branch attached to the cell body (fig. 1). The dendritic tree receives a set of $N$ weighted synaptic contacts from a set of afferent "axons". All synaptic contacts are excitatory. The output of the *clusteron* is given by

$$y = g(\sum_{i=1}^{N} a_i),\qquad(1)$$

where $a_i$ is the net excitatory input at synapse $i$ and $g$ is a thresholding nonlinearity. Unlike the TLU, in which the net input due to a single input line $i$ is $w_i x_i$, the net

input at a *clusteron* synapse $i$ with weight $w_i$ is given by,

$$a_i = w_i x_i \Big( \sum_{j \in \mathcal{D}_i} w_j x_j \Big), \tag{2}$$

where $x_i$ is the direct input stimulus intensity at synapse $i$, as for the TLU, and $\mathcal{D}_i = \{i-r, \ldots i, \ldots, i+r\}$ represents the neighborhood of radius $r$ around synapse $i$. It may be noted that the weight on each second-order term is constrained to be the product of elemental weights $w_i w_j$, such that the *clusteron* has only $N$ underlying degrees of freedom as compared to $N^2$ possible in a full second-order model. For the simplest case of $x_i \in \{0, 1\}$ and all weights set to 1, equation 2 says that the excitatory contribution of each active synapse is equal to the number of coactive synapses within its neighborhood. A synapse that is activated alone in its neighborhood thus provides a net excitatory input of $a_i = 1$; two synapses activated near to each other each provide a net excitatory input of $a_i = a_j = 2$, etc. The biophysical inspiration for the "multiplicative" relation in (2) is that, the net injected current through a region of voltage-dependent dendritic membrane can, under many circumstances, grow faster than linearly with increasing synaptic input to that region. Unlike the dendritic membrane modeled at the biophysical level, however, the *clusteron* in its current definition does not contain any saturating nonlinearities in the dendrites.

## 2.3   THE LEARNING PROBLEM

The learning problem of present interest is that of two-category classification. A pattern is a sparse $N$-element vector, where each component is a boolean random variable equal to 1 with probability $\rho$, and 0 otherwise. Let $T = \{t_1, t_2, \ldots, t_P\}$ be a training set consisting of $P$ randomly chosen patterns. The goal of the classifier is to respond with $y = 1$ to any pattern in $T$, and $y = 0$ to all other "control" patterns with the same average bit density $\rho$. Performance at this task is measured by the probability of correct classification on a test set consisting of equal numbers of training and control patterns.

## 2.4   THE LEARNING RULE

Learning in the *clusteron* is the process by which the ordering of synaptic connections onto the dendrite is manipulated. Second-order features that are statistically prominent in the training set, i.e. pairs of pattern components that are coactivated in the training set more often than average, can become encoded in the *clusteron* as pairs of synaptic connections within the same dendritic neighborhood.

Learning proceeds as follows. Each pattern in $T$ is presented once to the *clusteron* in a random sequence, constituting one training epoch. At the completion of each training epoch, each synapse $i$ whose activation averaged over the training set

$$<a_i> = \frac{1}{P} \sum_{p=1}^{P} a_i^{(p)}$$

falls below threshold $\theta$, is switched with another randomly chosen subthreshold synapse. The threshold can, for example, be chosen as $\theta = \frac{1}{N} \sum_{i=1}^{N} <a_i>$, i.e.

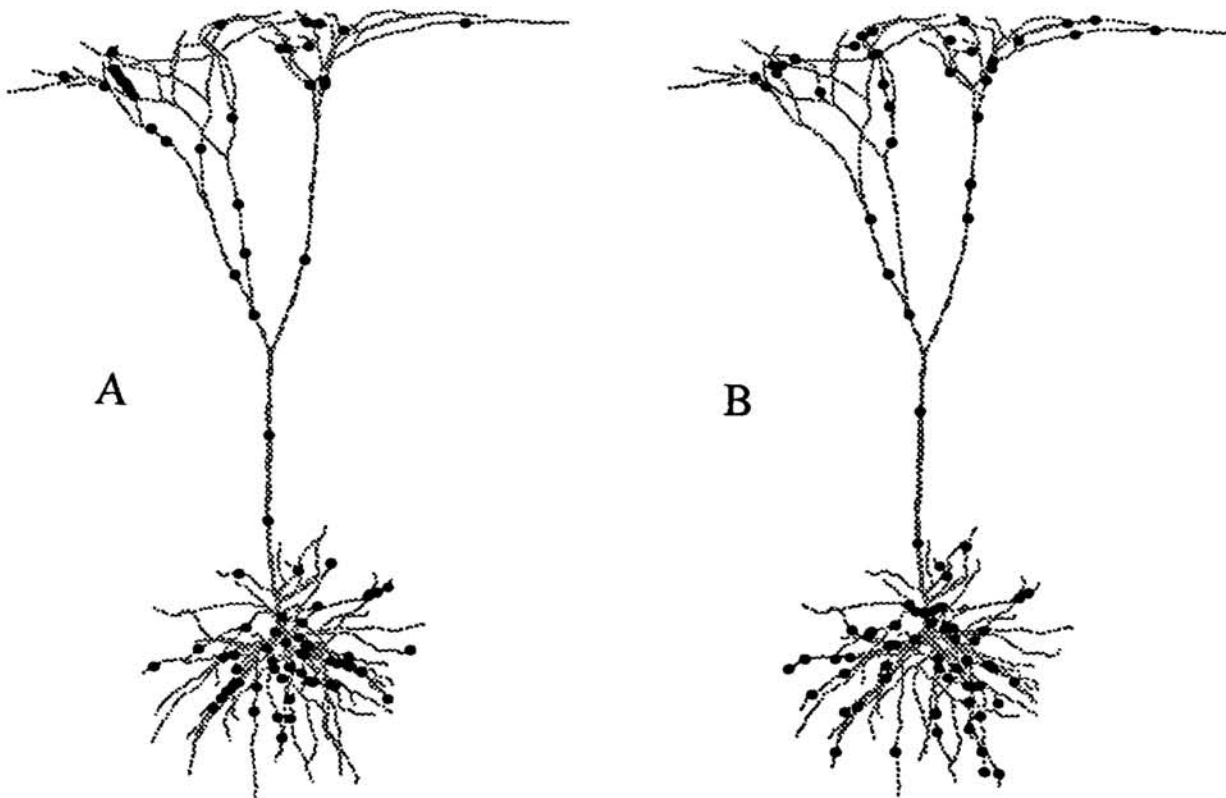

Figure 2: Distribution of 100 active synapses for a trained pattern (A) vs. a random control pattern (B); synapse locations are designated by black dots. Layout A is statistically more "clustery" than B, as evidenced by the presence of several clusters of 5 or more active synapses not found in B. While the total synaptic conductance activated in layout A was 20% less than that in layout B (linked to local variations in input-resistance), layout A generated 5 spikes at the soma, while layout B generated none.

the averaged synaptic activation across all synapses and training patterns. Each synapse whose average activation *exceeds* threshold $\theta$ is left undisturbed. Thus, if a synapse is often coactivated with its neighbors during learning, its average activation is high, and its connection is stabilized. If it is only rarely coactivated with its neighbors during learning, it loses its current connection, and is given the opportunity to stabilize a new connection at a new location.

The dynamics of *clusteron* learning may be caricatured as follows. At the start of learning, each "poor performing" synaptic connection improves its average activation level when switched to a new dendritic location where, by definition, it is expected to be an "average performer". The average global response $y$ to training patterns is thus also expected to increase during early training epochs. The average response to random controls remains unchanged, however, since there is no systematic structure in the ordering of synaptic connections relevant to any untrained pattern. This relative shift in the mean responses to training vs. control patterns is the basis for discrimination between them. The learning process approaches its asymptote as each pair of synapses switched, on average, disturbs the optimized *clusteron* neighborhood structure as much as it improves it.

## 3   RESULTS

The *clusteron* learning rule leads to a permutation of synaptic input connections having the property that the distribution of activated synapses in the dendritic tree associated with the presentation of a typical training pattern is statistically more "clustery" than the distribution of activated synapses associated with the presentation of a random control pattern.

For a given training set size, however, it is crucial to establish that the clustery distributions of active synapses associated with training patterns are in fact of a type that can be reliably discriminated—*within the detailed biophysical model*—from diffuse stimulation of the dendritic tree corresponding to unfamiliar stimulus patterns. In order to investigate this question, a *clusteron* with 17,000 synapses was trained with 1000 training patterns. This number of synapses was chosen in order that a direct map exist between *clusteron* synapses and dendritic spines, which were assumed to lie at 1 $\mu$m intervals along the approximately 17,000 $\mu$m of total dendritic length of the model neocortical neuron (from Douglas et al., 1991). In these runs, exactly 100 of the 17,000 bits were randomly set in each of the training and control patterns, such that every pattern activated exactly 100 synapses. After 200 training epochs, 100 training patterns and 100 control patterns were selected as a test set. For each test pattern, the locations of its 100 active *clusteron* synapses were mapped onto the dendritic tree in the biophysical model by traversing the latter in depth-first order. For example, training pattern #36 activated synapses as shown in fig. 2A, with synapse locations indicated by black dots. The layout in B was due to a control pattern. It may be perceived that layout A contains several clear groupings of 5 or more synapses that are not observed in layout B.

Within in the biophysical model, the conductance of each synapse, containing both NMDA and non-NMDA components, was scaled inversely with the input resistance measured locally at the dendritic spine head. Membrane parameters were similar to those used in (Mel, 1992); a high-threshold non-inactivating calcium conductance and an anomalous rectifier were used in these experiments as well, and were uniformly distributed over most of the dendritic tree. In the simulation run for each pattern, each of the 100 activated synapses was driven at 100 Hz for 100 ms, asynchronously, and the number of action potentials generated at the soma was counted. The total activated synaptic conductance in fig. 2A was 20% less than that activated by control layout B. However, layout A generated 5 somatic spikes while layout B generated none.

Fig. 3 shows the cell responses averaged over training patterns, four types of degraded training patterns, and control patterns. Most saliently, the average spike count in response to a training pattern was 3 times the average response to a control pattern. Not surprisingly, degraded training patterns gave rise to degraded responses. It is crucial to reiterate that all patterns, regardless of category, activated an identical number of synapses, with no average difference in their synaptic strengths or in dendritic eccentricity. Only the spatial distributions of active synapses were different among categories.

# 1000 Training Patterns

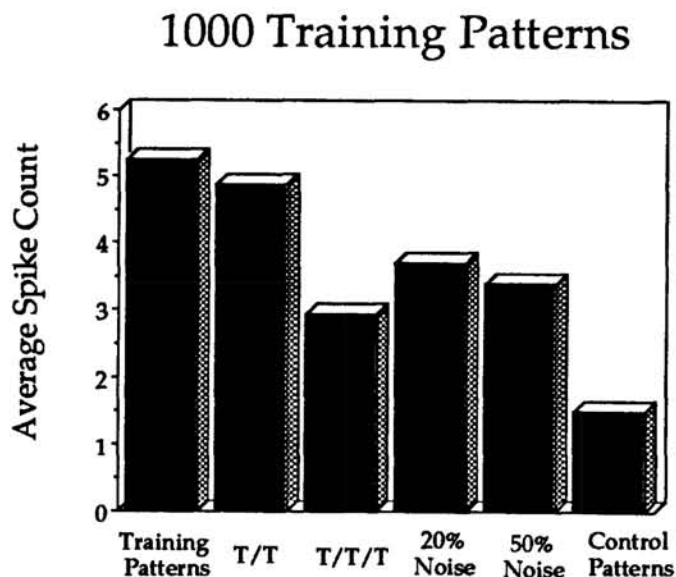

Figure 3: Average cell responses to training patterns, degraded training patterns, and control patterns. Categories designated T/T and T/T/T represented feature composites of 2 or 3 training patterns, respectively. Degraded responses to these categories of stimulus patterns was evidence for the underlying nonlinearity of the dendritic discriminant function.

## 4   CONCLUSION

These experiments within the *clusteron* model neuron have shown that the assumption of (1) dendritic cluster-sensitivity, (2) a combinatorially rich interface structure that allows every afferent axon potential access to many dendritic loci, and (3) a local Hebb-type learning rule for stabilizing newly formed synapses, are sufficient in principle to allow the learning of nonlinear input-ouput relations with a single dendritic tree. The massive rearrangement of synapses seen in these computational experiments is not strictly necessary; much of the work could be done instead through standard Hebbian synaptic potentiation, if a larger set of post-synaptic neurons is assumed to be available to each afferent instead of a single neuron as used here. Architectural issues relevant to this issue have been discussed at length in (Mel, 1990; Mel & Koch, 1990).

An analysis of the storage capacity of the *clusteron* will be presented elsewhere.

### Acknowledgements

This work was supported by the Office of Naval Research, the James McDonnell Foundation, and National Institute of Mental Health. Thanks to Christof Koch for providing an excellent working environment, Ken Miller for helpful discussions, and to Rodney Douglas for discussions and use of his neurons.

## References

Brown, T.H., Mainen, Z.F., Zador, A.M., & Claiborne, B.J. Self-organization of hebbian synapses in hippocampal neurons. In *Advances in Neural Information Processing Systems, vol. 3*, R. Lippmann, J. Moody, & D. Touretzky, (Eds.), Palo Alto: Morgan Kauffman, 1991.

Douglas, R.J., Martin, K.A.C., & Whitteridge, D. An intracellular analysis of the visual responses of neurones in striate visual cortex. *J. Physiol.*, 1991, *440*, 659-696.

Durbin, R. & Rumelhart, D.E. Product units: a computationally powerful and biologically plausible extension to backpropagation networks. *Neural Computation*, 1989, *1*, 133.

Feldman, J.A. & Ballard, D.H. Connectionist models and their properties. *Cognitive Science*, 1982, *6*, 205-254.

Hines, M. A program for simulation of nerve equations with branching geometries. *Int. J. Biomed. Comput.*, 1989, *24*, 55-68.

Mel, B.W. The sigma-pi column: a model for associative learning in cerebral neocortex. CNS Memo #6, Computation and Neural Systems Program, California Institute of Technology, 1990.

Mel, B.W. NMDA-based pattern classification in a modeled cortical neuron. 1992, *Neural Computation*, in press.

Mel, B.W. & Koch, C. Sigma-pi learning: On radial basis functions and cortical associative learning. In *Advances in neural information processing systems, vol. 2*, D.S. Touretzsky, (Ed.), San Mateo, CA: Morgan Kaufmann, 1990.